# Adjoint Operator Algorithms for Faster Learning in Dynamical Neural Networks

Jacob Barhen             Nikzad Toomarian             Sandeep Gulati

Center for Space Microelectronics Technology
Jet Propulsion Laboratory
California Institute of Technology
Pasadena, CA 91109

## ABSTRACT

A methodology for faster supervised learning in dynamical nonlinear neural networks is presented. It exploits the concept of *adjoint operators* to enable computation of changes in the network's response due to perturbations in all system parameters, using the solution of a single set of appropriately constructed linear equations. The lower bound on speedup per learning iteration over conventional methods for calculating the neuromorphic energy gradient is $O(N^2)$, where $N$ is the number of neurons in the network.

## 1  INTRODUCTION

The biggest promise of artifcial neural networks as computational tools lies in the hope that they will enable fast processing and synthesis of complex information patterns. In particular, considerable efforts have recently been devoted to the formulation of efficent methodologies for learning (e.g., Rumelhart et al., 1986; Pineda, 1988; Pearlmutter, 1989; Williams and Zipser, 1989; Barhen, Gulati and Zak, 1989). The development of learning algorithms is generally based upon the minimization of a neuromorphic energy function. The fundamental requirement of such an approach is the computation of the gradient of this objective function with respect to the various parameters of the neural architecture, e.g., synaptic weights, neural

gains, etc. The paramount contribution to the often excessive cost of learning using dynamical neural networks arises from the necessity to solve, at each learning iteration, one set of equations for each parameter of the neural system, since those parameters affect both directly and indirectly the network's energy.

In this paper we show that the concept of adjoint operators, when applied to dynamical neural networks, not only yields a considerable algorithmic speedup, but also puts on a firm mathematical basis prior results for "recurrent" networks, the derivations of which sometimes involved much heuristic reasoning. We have already used adjoint operators in some of our earlier work in the fields of energy-economy modeling (Alsmiller and Barhen, 1984) and nuclear reactor thermal hydraulics (Barhen et al., 1982; Toomarian et al., 1987) at the Oak Ridge National Laboratory, where the concept flourished during the past decade (Oblow, 1977; Cacuci et al., 1980).

In the sequel we first motivate and construct, in the most elementary fashion, a computational framework based on adjoint operators. We then apply our results to the Cohen-Grossberg-Hopfield (CGH) additive model, enhanced with terminal attractor (Barhen, Gulati and Zak, 1989) capabilities. We conclude by presenting the results of a few typical simulations.

## 2 ADJOINT OPERATORS

Consider, for the sake of simplicity, that a problem of interest is represented by the following system of $N$ coupled nonlinear equations

$$\bar{\varphi}(\bar{u}, \bar{p}) = 0 \qquad (2.1)$$

where $\bar{\varphi}$ denotes a nonlinear operator[1] . Let $\bar{u}$ and $\bar{p}$ represent the N-vector of dependent state variables and the M-vector of system parameters, respectively. We will assume that generally $M >> N$ and that elements of $\bar{p}$ are, in principle, independent. Furthermore, we will also assume that, for a specific choice of parameters, a unique solution of Eq. (2.1) exists. Hence, $\bar{u}$ is an implicit function of $\bar{p}$. A system "response", $R$, represents any result of the calculations that is of interest. Specifically

$$R = R(\bar{u}, \bar{p}) \qquad (2.2)$$

i.e., $R$ is a known nonlinear function of $\bar{p}$ and $\bar{u}$ and may be calculated from Eq. (2.2) when the solution $\bar{u}$ in Eq. (2.1) has been obtained for a given $\bar{p}$. The problem of interest is to compute the "sensitivities" of $R$, i.e., the derivatives of $R$ with respect to parameters $p_\mu$, $\mu = 1, \cdots, M$. By definition

$$\frac{dR}{dp_\mu} = \frac{\partial R}{\partial p_\mu} + \frac{\partial R}{\partial \bar{u}} \cdot \frac{\partial \bar{u}}{\partial p_\mu} \qquad (2.3)$$

Since the response $R$ is known analytically, the computation of $\partial R/\partial p_\mu$ and $\partial R/\partial \bar{u}$ is straightforward. The quantity that needs to be determined is the vector $\partial \bar{u}/\partial p_\mu$. Differentiating the state equations (2.1), we obtain a set of equations to be referred to as "forward" sensitivity equations

$$\frac{\partial \bar{\varphi}}{\partial \bar{u}} \cdot \frac{\partial \bar{u}}{\partial p_\mu} = -\frac{\partial \bar{\varphi}}{\partial p_\mu} \tag{2.4}$$

To simplify the notations, we are omitting the "transposed" sign and denoting the $N$ by $N$ forward sensitivity matrix $\partial \bar{\varphi}/\partial \bar{u}$ by $A$, the N-vector $\partial \bar{u}/\partial p_\mu$ by $^\mu \bar{q}$ and the "source" N-vector $-\partial \bar{\varphi}/\partial p_\mu$ by $^\mu \bar{s}$. Thus

$$A \, ^\mu \bar{q} = \, ^\mu \bar{s} \tag{2.5}$$

Since the source term in Eq. (2.5) explicitly depends on $\mu$, computing $dR/dp_\mu$, requires solving the above system of N algebraic equations for each parameter $p_\mu$. This difficulty is circumvented by introducing adjoint operators. Let $A^*$ denote the formal adjoint[2] of the operator A. The adjoint sensitivity equations can then be expressed as

$$A^* \, ^\mu \bar{q}^* = \, ^\mu \bar{s}^*. \tag{2.6}$$

By definition, for algebraic operators

$$^\mu \bar{q}^* \cdot (A \, ^\mu \bar{q}) = \, ^\mu \bar{q}^* \cdot \, ^\mu \bar{s} = \, ^\mu \bar{q} \cdot (A^* \, ^\mu \bar{q}^*) = \, ^\mu \bar{q} \cdot \, ^\mu \bar{s}^* \tag{2.7}$$

Since Eq. (2.3), can be rewritten as

$$\frac{dR}{dp_\mu} = \frac{\partial R}{\partial p_\mu} + \frac{\partial R}{\partial \bar{u}} \, ^\mu \bar{q} , \tag{2.8}$$

if we identify

$$\frac{\partial R}{\partial \bar{u}} \equiv \, ^\mu \bar{s}^* \equiv \bar{s}^* \tag{2.9}$$

we observe that the source term for the adjoint equations is independent of the specific parameter $p_\mu$. Hence, the *solution of a single set of adjoint equations will provide all the information required to compute the gradient of R with respect to all parameters.* To underscore that fact we shall denote $^\mu \bar{q}^*$ as $\bar{v}$. Thus

$$\frac{dR}{dp_\mu} = \frac{\partial R}{\partial p_\mu} - \bar{v} \cdot \frac{\partial \bar{\varphi}}{\partial p_\mu} \tag{2.10}$$

We will now apply this computational framework to a CGH network enhanced with terminal attractor dynamics. The model developed in the sequel differs from our

earlier formulations (Barhen, Gulati and Zak, 1989; Barhen, Zak and Gulati, 1989) in avoiding the use of constraints in the neuromorphic energy function, thereby eliminating the need for differential equations to evolve the concomitant Lagrange multipliers. Also, the usual activation dynamics is transformed into a set of equivalent equations which exhibit more "congenial" numerical properties, such as "contraction".

# 3  APPLICATIONS TO NEURAL LEARNING

We formalize a neural network as an adaptive dynamical system whose temporal evolution is governed by the following set of coupled nonlinear differential equations

$$\dot{z}_n + \kappa_n z_n = \sum_m \omega_{nm} T_{nm} g_\gamma(z_m) + {}^k I_n \qquad (3.1)$$

where $z_n$ represents the mean soma potential of the $n$th neuron and $T_{nm}$ denotes the synaptic coupling from the $m$-th to the $n$-th neuron. The weighting factor $\omega_{nm}$ enforces topological considerations. The constant $\kappa_n$ characterizes the decay of neuron activity. The sigmoidal function $g_\gamma(\cdot)$ modulates the neural response, with gain given by $\gamma_m$; typically, $g_\gamma(z) = \tanh(\gamma z)$. The "source" term ${}^k I_n$, which includes dimensional considerations, encodes contribution in terms of attractor coordinates of the $k$-th training sample via the following expression

$$
{}^k I_n = \begin{cases} [{}^k a_n]^{1-\beta} \, [{}^k a_n - g_\gamma(z_n)\,]^\beta & \text{if } n \in S_X \\ 0 & \text{if } n \in S_H \cup S_Y \end{cases} . \qquad (3.2)
$$

The topographic input, output and hidden network partitions $S_X$, $S_Y$ and $S_H$ are architectural requirements related to the encoding of mapping-type problems for which a number of possibilities exist (Barhen, Gulati and Zak, 1989; Barhen, Zak and Gulati, 1989). In previous articles (ibid; Zak, 1989) we have demonstrated that in general, for $\beta = (2i+1)^{-1}$ and $i$ a strictly positive integer, such attractors have infinite local stability and provide opportunity for learning in real-time. Typically, $\beta$ can be set to 1/3. Assuming an adiabatic framework, the fixed point equations at equilibrium, i.e., as $\dot{z}_n \to 0$, yield

$$\frac{\kappa_n}{\gamma_n} \, g^{-1}({}^k \tilde{u}_n) = \sum_m \omega_{nm} T_{nm} \, {}^k \tilde{u}_m + {}^k \tilde{I}_n \qquad (3.3)$$

where $u_n = g_\gamma(z_n)$ represents the neural response. The superscript $\sim$ denotes quantities evaluated at steady state. Operational network dynamics is then given by

$$\dot{u}_n + u_n = g_\gamma \left[ \frac{\gamma_n}{\kappa_n} \sum_m \omega_{nm} T_{nm} \, u_m + \frac{\gamma_n}{\kappa_n} \, {}^k I_n \right] \qquad (3.4)$$

To proceed formally with the development of a supervised learning algorithm, we consider an approach based upon the minimization of a constrained "neuromorphic" energy function $E$ given by the following expression

$$E(\bar{u}, \bar{p}) = \frac{1}{2} \sum_k \sum_n [\, {}^k \tilde{u}_n - {}^k a_n \,]^2 \qquad \forall \, n \in S_X \cup S_Y \qquad (3.5)$$

We relate adjoint theory to neural learning by identifying the neuromorphic energy function, E in Eq. (3.5), with the system response R. Also, let $\bar{p}$ denote the following system parameters:

$$\bar{p} = \{ T_{11}, \cdots T_{NN} \mid \kappa_1, \cdots \kappa_N \mid \gamma_1, \cdots \gamma_N \mid \cdots \}$$

The proposed objective function enforces convergence of every neuron in $S_X$ and $S_Y$ to attractor coordinates corresponding to the components in the input-output training patterns, thereby prompting the network to learn the embedded invariances. Lyapunov stability requires an energy-like function to be monotonically decreasing in time. Since in our model the internal dynamical parameters of interest are the synaptic strengths $T_{nm}$ of the interconnection topology, the characteristic decay constants $\kappa_n$ and the gain parameters $\gamma_n$ this implies that

$$\dot{E} = \sum_n \sum_m \frac{dE}{dT_{nm}} \dot{T}_{nm} + \sum_n \frac{dE}{d\kappa_n} \dot{\kappa}_n + \sum_n \frac{dE}{d\gamma_n} \dot{\gamma}_n < 0 \qquad (3.6)$$

For each adaptive system parameter, $p_\mu$, Lyapunov stability will be satisfied by the following choice of equations of motion

$$\dot{p}_\mu = -\tau_p \frac{dE}{dp_\mu} \qquad (3.7)$$

Examples include

$$\dot{T}_{nm} = -\tau_T \frac{dE}{dT_{nm}} \qquad ; \qquad \dot{\gamma}_n = -\tau_\gamma \frac{dE}{d\gamma_n} \qquad ; \qquad \dot{\kappa}_n = -\tau_\kappa \frac{dE}{d\kappa_n}$$

where the time-scale parameters $\tau_T$, $\tau_\kappa$ and $\tau_\gamma > 0$. Since E depends on $p_\mu$ both directly and indirectly, previous methods required solution of a system of $N$ equations for each parameter $p_\mu$ to obtain $dE/dp_\mu$ from $d\tilde{u}/dp_\mu$. Our methodology (based on **adjoint operators**), yields all derivatives $dE/dp_\mu, \forall \mu$, by solving a single set of $N$ linear equations.

The nonlinear neural operator for each training pattern $k$, $k = 1, \cdots K$, at equilibrium is given by

$$^k\varphi_n \, (^k\tilde{u}, \bar{p}) = g \left[ \frac{1}{\kappa_n} \sum_{m'} \omega_{nm'} \, T_{nm'} \, {}^k\tilde{u}_{m'} + \frac{1}{\kappa_n} \, {}^k\tilde{I}_n \right] - {}^k\tilde{u}_n = 0 \qquad (3.8)$$

where, without loss of generality we have set $\gamma_n$ to unity. So, in principle ${}^k\tilde{u}_n = {}^k\tilde{u}_n [T, \bar{\kappa}, \bar{\gamma}, {}^k a_n, \cdots]$. Using Eqs. (3.8), the forward sensitivity matrix can be computed and compactly expressed as

$$
\begin{aligned}
^k A_{nm} = \frac{\partial \, ^k\varphi_n}{\partial \, ^k\tilde{u}_m} &= {}^k\hat{g}_n \frac{1}{\kappa_n} \left[ \omega_{nm} \, T_{nm} + \frac{\partial \, ^k\tilde{I}_n}{\partial \, ^k\tilde{u}_m} \right] - \delta_{nm} \\
&= \frac{1}{\kappa_n} \, ^k\hat{g}_n \, \omega_{nm} \, T_{nm} - {}^k\eta_n \, \delta_{nm}.
\end{aligned}
\qquad (3.9)
$$

where

$$
{}^k\eta_n \;=\; \begin{cases} 1 + \frac{[{}^k a_n]^{2/3}}{3\kappa_n}\,{}^k\hat{g}_n\,[{}^k a_n - {}^k\tilde{u}_n\,]^{-2/3} & \text{if } n \in S_X \\ 1 & \text{if } n \in S_H \cup S_Y \end{cases} . \tag{3.10}
$$

Above, ${}^k\hat{g}_n$ represents the derivative of $g$ with respect to ${}^k\tilde{u}_n$, i.e., if $g \equiv tanh$, then

$$
{}^k\hat{g}_n \;=\; 1 - [{}^k g_n]^2 \quad \text{where} \quad {}^k g_n \;=\; g\Big[\frac{1}{\kappa_n}\Big(\sum_m \omega_{nm}\,T_{nm}\,{}^k\tilde{u}_m + {}^k\tilde{I}_n\Big)\Big] \tag{3.11}
$$

Recall that the formal adjoint equation is given as $A^*\bar{v} = \bar{s}^*$ ; here

$$
{}^k A^*_{nm} \;=\; \frac{1}{\kappa_m}\,{}^k\hat{g}_m\,\omega_{mn}\,T_{mn} \;-\; {}^k\eta_m\,\delta_{mn} \tag{3.12}
$$

Using Eqs. (2.9) and (3.5), we can compute the formal **adjoint source**

$$
{}^k s^*_n \;\equiv\; \frac{\partial E}{\partial\,{}^k\tilde{u}_n} \;=\; \begin{cases} {}^k\tilde{u}_n - {}^k a_n & \text{if } n \in S_X \cup S_Y \\ 0 & \text{if } n \in S_H \end{cases} \tag{3.13}
$$

The system of adjoint fixed-point equations can then be constructed using Eqs. (3.12) and (3.13), to yield :

$$
\sum_m \frac{1}{\kappa_m}\,{}^k\hat{g}_m\,\omega_{mn}\,T_{mn}\,{}^k\tilde{v}_m \;-\; \sum_m {}^k\eta_m\,\delta_{mn}\,{}^k\tilde{v}_m \;=\; {}^k s^*_n \tag{3.14}
$$

Notice that the above coupled system, (3.14), is linear in ${}^k\tilde{v}$. Furthermore, it has the same mathematical characteristics as the operational dynamics (3.4). Its components can be obtained as the equilibrium points, (i.e., $\dot{v}_i \to 0$) of the **adjoint neural dynamics**

$$
\dot{v}_n + {}^k\eta_n\,v_n \;=\; \sum_m \frac{1}{\kappa_m}\,{}^k\hat{g}_m\,\omega_{mn}\,T_{mn}\,v_m \;-\; {}^k s^*_n \tag{3.15}
$$

As an implementation example, let us conclude by deriving the learning equations for the synaptic strengths, $T_\mu$. Recall that

$$
\frac{dE}{dT_\mu} \;=\; \frac{\partial E}{\partial T_\mu} \;+\; \sum_k {}^k\tilde{v}\,\cdot\,{}^{\mu k}\bar{s} \qquad \mu = (i,j) \tag{3.16}
$$

We differentiate the steady state equations (3.8) with respect to $T_{ij}$, to obtain the forward source term,

$$
{}^{\mu k} s_n \;=\; -\frac{\partial\,{}^k\varphi_n}{\partial T_{ij}} \;=\; -{}^k\hat{g}_n\Big[\frac{1}{\kappa_n}\sum_l \omega_{nl}\,\delta_{in}\,\delta_{jl}\,{}^k\tilde{u}_l + 0\Big]
$$

$$
\;=\; -\frac{1}{\kappa_n}\,{}^k\hat{g}_n\,\delta_{in}\,\omega_{nj}\,{}^k\tilde{u}_j \tag{3.17}
$$

Since by definition, $\partial E / \partial T_{nm} = 0$ , the explicit energy gradient contribution is obtained as

$$\dot{T}_{nm} = -\tau_T \left[ -\frac{\omega_{nm}}{\kappa_n} \sum_k {}^k\tilde{v}_n \, {}^k\hat{g}_n \, {}^k\tilde{u}_m \right] \tag{3.18}$$

It is straightforward to obtain learning equations for $\gamma_n$ and $\kappa_n$ in a similar fashion.

## 4  ADAPTIVE TIME-SCALES

So far the adaptive learning rates, i.e., $\tau_p$ in Eq.(3.7), have not been specified. Now we will show that, by an appropriate selection of these parameters the convergence of the corresponding dynamical systems can be considerably improved. Without loss of generality, we shall assume $\tau_T = \tau_\kappa = \tau_\gamma = \tau$, and we shall seek $\tau$ in the form (Barhen et al, 1989; Zak 1989)

$$\tau \propto |\nabla E|^{-\beta} \tag{4.1}$$

where $\nabla E$ denotes the vector with components $\nabla_T E$, $\nabla_\gamma E$ and $\nabla_\kappa E$. It is straightforward to show that

$$\frac{d}{dt} |\nabla E| = -\chi |\nabla E|^{1-\beta} \tag{4.2}$$

as $\nabla E$ tends to zero, where $\chi$ is an arbitrary positive constant. If we evaluate the relaxation time of the energy gradient, we find that

$$t_E = \int_{|\nabla E|_0}^{|\nabla E| \to 0} \frac{d |\nabla E|}{|\nabla E|^{1-\beta}} = \begin{cases} \infty & \text{if } \beta \le 0 \\ \frac{1}{\beta} |\nabla E|_0^\beta < \infty & \text{if } \beta > 0 \end{cases} \tag{4.3}$$

Thus, for $\beta \le 0$ the relaxation time is infinite, while for $\beta > 0$ it is finite. The dynamical system (3.19) suffers a qualitative change for $\beta > 0$ : it loses uniqueness of solution. The equilibrium point $|\nabla E| = 0$ becomes a singular solution being intersected by all the transients, and the Lipschitz condition is violated, as one can see from

$$\frac{d}{d |\nabla E|} \left( \frac{d |\nabla E|}{dt} \right) = -\chi |\nabla E|^{-\beta} \longrightarrow -\infty \tag{4.4}$$

where $|\nabla E|$ tends to zero, while $\beta$ is strictly positive. Such infinitely stable points are "terminal attractors". By analogy with our previous results we choose $\beta = 2/3$, which yields

$$\tau = \left( \sum_n \sum_m [\nabla_T E]_{nm}^2 + \sum_n [\nabla_\gamma E]_n^2 + \sum_n [\nabla_\kappa E]_n^2 \right)^{-1/3} \tag{4.5}$$

The introduction of these adaptive time-scales dramatically improves the convergence of the corresponding learning dynamical systems.

## 5   SIMULATIONS

The computational framework developed in the preceding section has been applied to a number of problems that involve learning nonlinear mappings, including Exclusive-OR, the hyperbolic tangent and trignometric functions, e.g., sin. Some of these mappings (e.g., XOR) have been extensively benchmarked in the literature, and provide an adequate basis for illustrating the computational efficacy of our proposed formulation. Figures 1(a)-1(d) demonstrate the temporal profile of various network elements during learning of the XOR function. A six neuron feedforward network was used, that included self-feedback on the output unit and bias. Fig. 1(a) shows the LMS error during the training phase. The worst-case convergence of the output state neuron to the presented attractor is displayed in Fig. 1(b). Notice the rapid convergence of the input state due to the terminal attractor effect. The behavior of the adaptive time-scale parameter $\tau$ is depicted in Fig. 1(c). Finally, Fig. 1(d) shows the evolution of the energy gradient components.

The test setup for signal processing applications, i.e., learning the sin function and the tanh sigmoidal nonlinearlity, included a 8-neuron fully connected network with no bias. In each case the network was trained using as little as 4 randomly sampled training points. Efficacy of recall was determined by presenting 100 random samples. Fig. (2) and (3b) illustrate that we were able to approximate the sin and the hyperbolic tangent functions using 16 and 4 pairs respectively. Fig. 3(a) demonstrates the network performance when 4 pairs were used to learn the hyperbolic tangent.

We would like to mention that since our learning methodology involves terminal attractors, extreme caution must be exercised when simulating the algorithms in a digital computing environment. Our discussion on sensitivity of results to the integration schemes (Barhen, Zak and Gulati, 1989) emphasizes that explicit methods such as Euler or Runge-Kutta shall not be used, since the presence of terminal attractors induces extreme stiffness. Practically, this would require an integration time-step of infinitesimal size, resulting in numerical round-off errors of unacceptable magnitude. Implicit integration techniques such as the Kaps-Rentrop scheme should therefore be used.

## 6   CONCLUSIONS

In this paper we have presented a theoretical framework for faster learning in dynamical neural networks. Central to our approach is the concept of *adjoint operators* which enables computation of network neuromorphic energy gradients with respect to all system parameters using the solution of a single set of linear equations. If $C_F$ and $C_A$ denote the computational costs associated with solving the forward and adjoint sensitivity equations (Eqs. 2.5 and 2.6), and if $M$ denotes the number of parameters of interest in the network, the speedup achieved is

$$S^{F \to A} \;=\; \frac{M\,C_F}{C_A}$$

If we assume that $C_F \simeq C_A$ and that $M = N^2 + 2N + \cdots$, we see that the lower bound on speedup per learning iteration is $O(N^2)$. Finally, particular care must be execrcised when integrating the dynamical systems of interest, due to the extreme stiffness introduced by the terminal attractor constructs.

## Acknowledgements

The research described in this paper was performed by the Center for Space Microelectronics Technology, Jet Propulsion Laboratory, California Institute of Technology, and was sponsored by agencies of the U.S. Department of Defense, and by the Office of Basic Energy Sciences of the U.S. Department of Energy, through interagency agreements with NASA.

## Footnotes

[1] If differential operators appear in Eq. (2.1), then a corresponding set of boundary and/or initial conditions to specify the domain of $\varphi$ must also be provided. In general an inhomogeneous "source" term can also be present. The learning model discussed in this paper focuses on the adiabatic approximation only. Nonadiabatic learning algorithms, wherein the response is defined as a functional, will be discussed in a forthcoming article.

[2] Adjoint operators can only be considered for densely defined linear operators on Banach spaces (see e.g., Cacuci, 1980). For the neural application under consideration we will limit ourselves to real Hilbert spaces. Such spaces are self-dual. Furthermore, the domain of an adjoint operator is determined by selecting appropriate adjoint boundary conditions[1]. The associated bilinear form evaluated on the domain boundary must thus be also generally included.

## References

R.G. Alsmiller, J. Barhen and J. Horwedel. (1984) "The Application of Adjoint Sensitivity Theory to a Liquid Fuels Supply Model", *Energy*, 9(3), 239-253.

J. Barhen, D.G. Cacuci and J.J. Wagschal. (1982) "Uncertainty Analysis of Time-Dependent Nonlinear Systems", *Nucl. Sci. Eng.*, 81, 23-44.

J. Barhen, S. Gulati and M. Zak. (1989) "Neural Learning of Constrained Nonlinear Transformations", *IEEE Computer*, 22(6), 67-76.

J. Barhen, M. Zak and S. Gulati. (1989) " Fast Neural Learning Algorithms Using Networks with Non-Lipschitzian Dynamics", in *Proc. Neuro-Nimes '89*, 55-68, EC2, Nanterre, France.

D.G. Cacuci, C.F. Weber, E.M. Oblow and J.H. Marable. (1980) "Sensitivity Theory for General Systems of Nonlinear Equations", *Nucl. Sci. Eng.*, 75, 88-110.

E.M. Oblow. (1977) "Sensitivity Theory for General Non-Linear Algebraic Equations with Constraints", ORNL/TM-5815, Oak Ridge National Laboratory.

B.A. Pearlmutter. (1989) "Learning State Space Trajectories in Recurrent Neural Networks", *Neural Computation*, 1(3), 263-269.

F.J. Pineda. (1988) "Dynamics and Architecture in Neural Computation", *Journal of Complexity*, 4, 216-245.

D.E. Rumelhart and J.L. Mclelland. (1986) *Parallel and Distributed Procesing*, MIT Press, Cambridge, MA.

N. Toomarian, E. Wacholder and S. Kaizerman. (1987) "Sensitivity Analysis of Two-Phase Flow Problems", *Nucl. Sci. Eng.*, 99(1), 53-81.

R.J. Williams and D. Zipser. (1989) "A Learning Algorithm for Continually Running Fully Recurrent Neural Networks", *Neural Computation*, 1(3), 270-280.

M. Zak. (1989) "Terminal Attractors", *Neural Networks*, 2(4), 259-274.

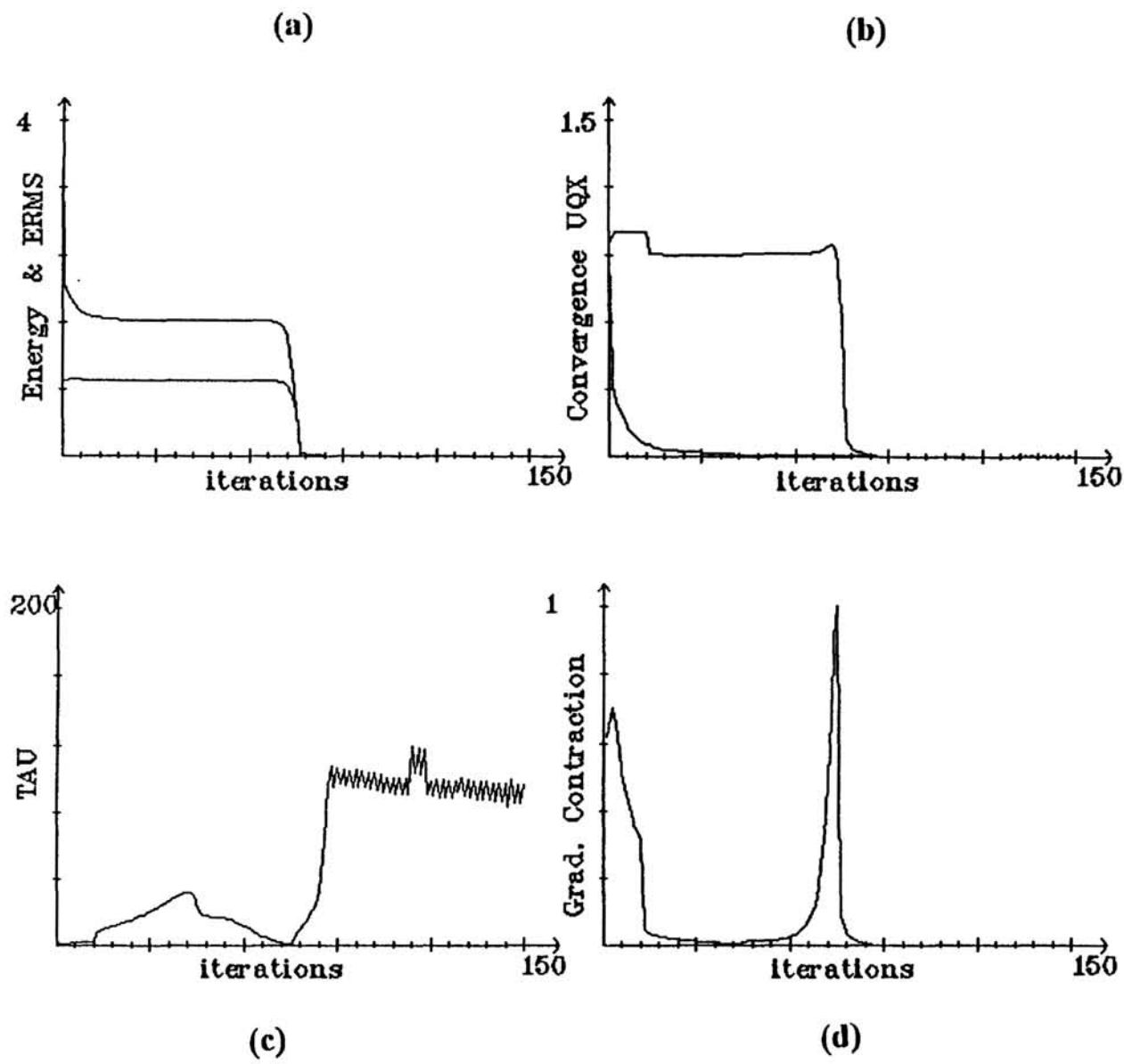

Figure 1(a)-(d).     Learning the Exclusive-OR function using a 6-neuron (including bias) feedforward dynamical network with self-feedback on the output unit.

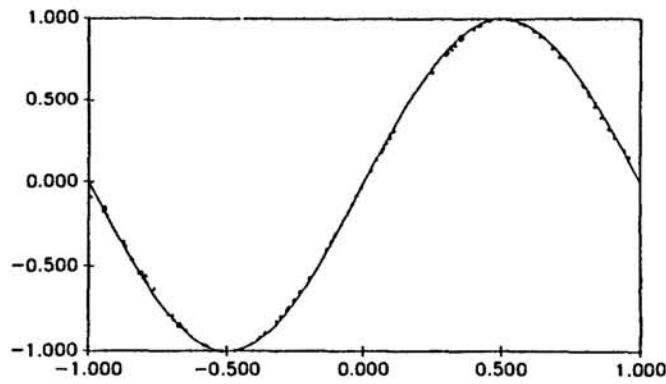

Figure 2.    Learning the Sin function using a fully connected, 8-neuron network with no bias. The training set comprised of 4 points that were randomly selected.

3(a)

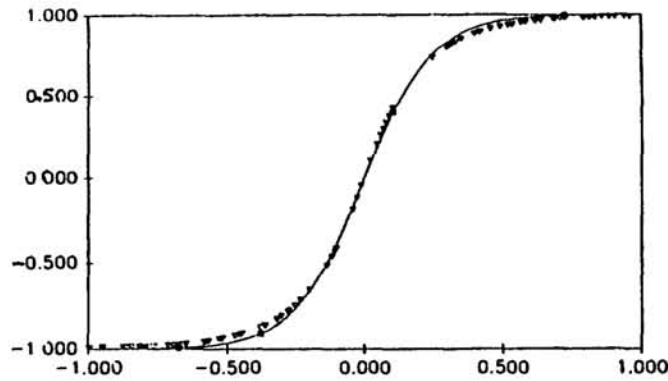

3(b)

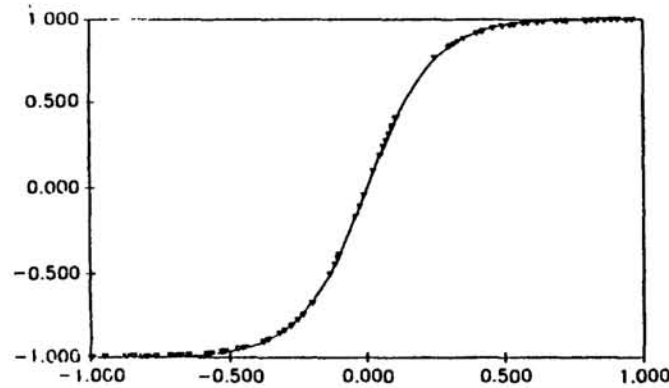

Figure 3.    Learning the Hyperbolic Tangent function using a fully connected, 8-neuron network with no bias. (a) using 4 randomly selected training samples; (b) using 16 randomly selected training samples.